# A Temporal Kernel-Based Model for Tracking Hand-Movements from Neural Activities

**Lavi Shpigelman**[12] **Koby Crammer**[1] **Rony Paz**[23] **Eilon Vaadia**[23] **Yoram Singer**[1]

[1] School of computer Science and Engineering
[2] Interdisciplinary Center for Neural Computation
[3] Dept. of Physiology, Hadassah Medical School
The Hebrew University Jerusalem, 91904, Israel
Email for correspondance: shpigi@cs.huji.ac.il

## Abstract

We devise and experiment with a dynamical kernel-based system for tracking hand movements from neural activity. The state of the system corresponds to the hand location, velocity, and acceleration, while the system's input are the instantaneous spike rates. The system's state dynamics is defined as a combination of a linear mapping from the previous *estimated* state and a kernel-based mapping tailored for modeling neural activities. In contrast to generative models, the activity-to-state mapping is learned using discriminative methods by minimizing a noise-robust loss function. We use this approach to predict hand trajectories on the basis of neural activity in motor cortex of behaving monkeys and find that the proposed approach is more accurate than both a static approach based on support vector regression and the Kalman filter.

## 1  Introduction

The paper focuses on the problem of tracking hand movements, which constitute smooth spatial trajectories, from spike trains of a neural population. We do so by devising a dynamical system which employs a tailored kernel for spike trains along with a linear mapping corresponding to the states' dynamics. Consider a situation where a subject performs free hand movements during a task that requires accurate space and time precision. In the lab, it may be a constrained reaching task while in real life it may be an every day task such as eating. We wish to track the hand position given only spike trains from a recorded neural population. The rationale of such an undertaking is two fold. First, this task can be viewed as a stem towards the development of a Brain Machine Interface (BMI) which gradually and rapidly become a possible future solution for the motor disabled patients. Recent studies of BMIs [13, 3, 10] (being on-line and feedback enabled) show that a relatively small number of cortical units can be used to move a cursor or a robot effectively, even without generation of hand movements and that training of the subjects improves the overall success of the BMIs. Second, an open loop (off-line) movement decoding (see e.g. [7, 1, 15, 11, 8]), while inappropriate for BMIs, is computationally less expensive, easier to implement and allows repeated analysis thus providing a handle to understandings of neural computations in the brain.

Early studies [6] showed that the direction of arm movement is reflected by the population vector of preferred directions weighted by current firing rates, suggesting that intended

movement is encoded in the firing rate which, in turn, is modulated by the angle between a unit's preferred direction (PD) and the intended direction. This linear regression approach is still prevalent and is applied, with some variation of the learning methods, in closed and open loop settings. There is relatively little work on the development of dedicated nonlinear methods.

Both movement and neural activity are dynamic and can therefore be naturally modeled by dynamical systems. Filtering methods often employ generative probabilistic models such as the well known Kalman filter [16] or more neurally specialized models [1] in which a cortical unit's spike count is generated by a probability function of its underlying firing rate which is tuned to movement parameters. The movement, being a smooth trajectory, is modeled as a linear transition with (typically additive Gaussian) noise. These methods have the advantage of being aware of the smooth nature of movement and provide models of what neurons are tuned to. However, the requirement of describing a neural population's firing probability as a function of movement state is hard to satisfy without making costly assumptions. The most prominent is the assumption of statistical independence of cells given the movement.

Kernel based methods have been shown to achieve state of the art results in many application domains. Discriminative kernel methods, such as Support Vector Regression (SVR) forgo the task of modeling neuronal tuning functions. Furthermore, the construction of kernel induced feature spaces, lends itself to efficient implementation of distance measures over spike trains that are better suited to comparing two neural population trajectories than the Euclidean distance in the original space of spike counts per bins [11, 5]. However, SVR is a "static" method that does not take into account the smooth dynamics of the predicted movement trajectory which imposes a statistical dependency between consecutive examples.

This paper introduces a kernel based regression method that incorporates linear dynamics of the predicted trajectories. In Sec. 2 we formally describe the problem setting. We introduce the movement tracking model and the associated learning framework in Sec. 3. The resulting learning problem yields a new kernel for linear dynamical systems. We provide an efficient calculation of this kernel and describe our dual space optimization method for solving the learning problem. The experimental method is presented in Sec. 4. Results, underscoring the merits of our algorithm are provided in Sec. 5 and conclusions are given in Sec. 6.

## 2   Problem Setting

Our training set contains $m$ trials. Each trial (typically indexed by $i$ or $j$) consists of a pair of movement and neural recordings, designated by $\left\{ \mathbf{Y}^i, \mathbf{O}^i \right\}$. $\mathbf{Y}^i = \left\{ \mathbf{y}_t^i \right\}_{t=1}^{t_{end}^i}$ is a time series of movement state values and $\mathbf{y}_t^i \in \mathbf{R}^d$ is the movement state vector at time $t$ in trial $i$. We are interested in reconstructing position, however, for better modeling, $\mathbf{y}_t^i$ may be a vector of position, velocity and acceleration (as is the case in Sec. 4). This trajectory is observed during model learning and is the inference target. $\mathbf{O}^i = \left\{ \mathbf{o}_t \right\}_{t=1}^{t_{end}^i}$ is a time series of neural spike counts and $\mathbf{o}_t^i \in \mathbf{R}^q$ is a vector of spike counts from $q$ cortical units at time $t$. We wish to learn a function $\mathbf{z}_t^i = f\left( \mathbf{O}_{1:t}^i \right)$ that is a good estimate (in a sense formalized in the sequel) of the movement $\mathbf{y}_t^i$. Thus, $f$ is a causal filtering method.

We confine ourselves to a causal setting since we plan to apply the proposed method in a closed loop scenario where real-time output is required. The partition into separate trajectories is a natural one in a setting where a session is divided into many trials, each consisting of one attempt at accomplishing the basic task (such as reaching movements to displayed targets). In tasks that involve no hitting of objects, hand movements are typically smooth.

End point movement in small time steps is loosely approximated as having constant acceleration. On the other hand, neural spike counts (which are typically measured in bins of $50 - 100ms$) vary greatly from one time step to the next. In summary, our goal is to devise a dynamic mapping from sequences of neural activities ending at a given time to the instantaneous hand movement characterization (location, velocity, and acceleration).

## 3 Movement Tracking Algorithm

Our regression method is defined as follows: given a series $\mathbf{O} \in \mathrm{R}^{q \times t_{end}}$ of observations and, possibly, an initial state $\mathbf{y}_0$, the predicted trajectory $\mathbf{Z} \in \mathrm{R}^{d \times t_{end}}$ is,

$$\mathbf{z}_t = \mathbf{A}\mathbf{z}_{t-1} + \mathbf{W}\phi\left(\mathbf{o}_t\right) \quad , t_{end} \geq t > 0, \tag{1}$$

where $\mathbf{z}_0 = \mathbf{y}_0$, $\mathbf{A} \in \mathrm{R}^{d \times d}$ is a matrix describing linear movement dynamics and $\mathbf{W} \in \mathrm{R}^{d \times q}$ is a weight matrix. $\phi\left(\mathbf{o}_t\right)$ is a feature vector of the observed spike trains at time $t$ and is later replaced by a kernel operator (in the dual formulation to follow). Thus, the state transition is a linear transformation of the previous state with the addition of a non-linear effect of the observation.

Note that unfolding the recursion in Eq. (1) yields $\mathbf{z}_t = \mathbf{A}^t \mathbf{y}_0 + \sum_{k=1}^{t} \left( \mathbf{A}^{t-k} \mathbf{W}\phi\left(\mathbf{o}_k\right)\right)$. Assuming that $\mathbf{A}$ describes stable dynamics (the real parts of the eigenvalues of $\mathbf{A}$ are les than 1), then the current prediction depends, in an exponentially decaying manner, on the previous observations. We further assume that $\mathbf{A}$ is fixed and wish to learn $\mathbf{W}$ (we describe our choice of $\mathbf{A}$ in Sec. 4). In addition, $\mathbf{o}_t$ may also encompass a series of previous spike counts in a window ending at time $t$ (as is the case in Sec. 4). Also, note that this model (in its non-kernelized version) has an algebraic form which is similar to the Kalman filter (to which we compare our results later).

**Primal Learning Problem:** The optimization problem presented here is identical to the standard SVR learning problem (see, for example [12]) with the exception that $\mathbf{z}_t^i$ is defined as in Eq. (1) while in standard SVR, $\mathbf{z}_t = \mathbf{W}\phi\left(\mathbf{o}_t\right)$ (i.e. without the linear dynamics). Given a training set of fully observed trials $\left\{\mathbf{Y}^i, \mathbf{O}^i\right\}_{i=1}^{m}$ we define the learning problem to be

$$\min_{\mathbf{W}} \quad \frac{1}{2}\left\|\mathbf{W}\right\|^2 + c \sum_{i=1}^{m} \sum_{t=1}^{t_{end}^i} \sum_{s=1}^{d} \left| \left(\mathbf{z}_t^i\right)_s - \left(\mathbf{y}_t^i\right)_s \right|_\varepsilon . \tag{2}$$

Where $\left\|\mathbf{W}\right\|^2 = \sum_{a,b} \left(\mathbf{W}\right)_{ab}^2$ (is the Forbenius norm). The second term is a sum of training errors (in all trials, times and movement dimensions). $\left| \cdot \right|_\varepsilon$ is the $\varepsilon$ insensitive loss and is defined as $\left|v\right|_\varepsilon = \max\left\{0, \left|v\right| - \varepsilon\right\}$. The first term is a regularization term that promotes small weights and $c$ is a fixed constant providing a tradeoff between the regularization term and the training error. Note that to compensate for different units and scales of the movement dimensions one could either define a different $\varepsilon_s$ and $c_s$ for each dimension of the movement or, conversely, scale the $s^{\text{th}}$ movement dimension. The tracking method, combined with the optimization specified here, defines the complete algorithm. We name this method the Discriminative Dynamic Tracker or DDT in short.

**A Dual Solution:** The derivation of the dual of the learning problem defined in Eq. (2) is rather mundane (e.g. [12]) and is thus omitted. Briefly, we replace the $\varepsilon$-loss with pairs of slack variables. We then write a Lagrangian of the primal problem and replace $\mathbf{z}_t^i$ with its (less-standard) definition. We then differentiate the Lagrangian with respect to the slack variables and $\mathbf{W}$ and obtain a dual optimization problem. We present the dual dual problem in a top-down manner, starting with the general form and finishing with a kernel definition. The form of the dual is

$$\max_{\boldsymbol{\alpha}, \boldsymbol{\alpha}^*} \quad -\frac{1}{2}\left(\boldsymbol{\alpha}^* - \boldsymbol{\alpha}\right)^T \mathcal{G}\left(\boldsymbol{\alpha}^* - \boldsymbol{\alpha}\right) + \left(\boldsymbol{\alpha}^* - \boldsymbol{\alpha}\right)^T \mathbf{y} - \left(\boldsymbol{\alpha}^* + \boldsymbol{\alpha}\right)^T \boldsymbol{\epsilon}$$

$$s.t. \; \boldsymbol{\alpha}, \boldsymbol{\alpha}^* \in [\mathbf{0}, \mathbf{c}] \qquad\qquad . \tag{3}$$

Note that the above expression conforms to the dual form of SVR. Let $\ell$ equal the size of the movement space ($d$), multiplied by the total number of time steps in all the training trajectories. $\boldsymbol{\alpha}, \boldsymbol{\alpha}^* \in \mathbf{R}^\ell$ are vectors of Lagrange multipliers, $\mathbf{y} \in \mathbf{R}^\ell$ is a column concatenation of all the training set movement trajectories $\left[\left(\mathbf{y}_1^1\right)^T \cdots \left(\mathbf{y}_{t_{end}^m}^m\right)^T\right]^T$, $\boldsymbol{\epsilon} = [\varepsilon, \ldots, \varepsilon]^T \in \mathbf{R}^\ell$ and $\mathcal{G} \in \mathbf{R}^{\ell \times \ell}$ is a Gram matrix ($\mathbf{v}^T$ denotes transposition). One obvious difference between our setting and the standard SVR lies within the size of the vectors and Gram matrix. In addition, a major difference is the definition of $\mathcal{G}$. We define $\mathcal{G}$ here in a hierarchical manner. Let $i, j \in \{1, \ldots, m\}$ be trajectory (trial) indexes. $\mathcal{G}$ is built from blocks indexed by $\mathbf{G}^{ij}$, which are in turn made from basic blocks, indexed by $\mathbf{K}_{tq}^{ij}$ as follows

$$\mathcal{G} = \begin{pmatrix} \mathbf{G}^{11} & \cdots & \mathbf{G}^{1m} \\ \vdots & \ddots & \vdots \\ \mathbf{G}^{m1} & \cdots & \mathbf{G}^{mm} \end{pmatrix} \quad , \quad \mathbf{G}^{ij} = \begin{pmatrix} \mathbf{K}_{11}^{ij} & \cdots & \mathbf{K}_{1t_j}^{ij} \\ \vdots & \ddots & \vdots \\ \mathbf{K}_{t_{end}^i 1}^{ij} & \cdots & \mathbf{K}_{t_{end}^i t_{end}^j}^{ij} \end{pmatrix},$$

where block $\mathbf{G}^{ij}$ refers to a pair of trials ($i$ and $j$). Finally Each basic block, $\mathbf{K}_{tq}^{ij}$ refers to a pair of time steps $t$ and $q$ in trajectories $i$ and $j$ respectively. $t_{end}^i, t_{end}^j$ are the time lengths of trials $i$ and $j$. Basic blocks are defined as

$$\mathbf{K}_{tq}^{ij} = \sum_{r=1}^{t} \sum_{s=1}^{q} \left(\mathbf{A}^{t-r}\right) k_{rs}^{ij} \left(\mathbf{A}^{q-s}\right)^T , \tag{4}$$

where $k_{rs}^{ij} = k\left(\mathbf{o}_r^i, \mathbf{o}_s^j\right)$ is a (freely chosen) basic kernel between the two neural observations $\mathbf{o}_r^i$ and $\mathbf{o}_s^j$ at times $r$ and $s$ in trials $i$ and $j$ respectively. For an explanation of kernel operators we refer the reader to [14] and mention that the kernel operator can be viewed as computing $\phi\left(\mathbf{o}_r^i\right) \cdot \phi\left(\mathbf{o}_s^j\right)$ where $\phi$ is a fixed mapping to some inner product space. The choice of kernel (being the choice of feature space) reflects a modeling decision that specifies how similarities between neural patterns are measured. The resulting dual form of the tracker is $\mathbf{z}_t = \sum_k \boldsymbol{\alpha}_k \mathcal{G}_{tk}$ where $\mathcal{G}_t$ is the Gram matrix row of the new example.

It is therefore clear from Eq. (4) that the linear dynamic characteristics of DDT results in a Gram matrix whose entries depend on previous observations. This dependency is exponentially decaying as the time difference between events in the trajectories grow. Note that solution of the dual optimization problem in Eq. (3) can be calculated by any standard quadratic programming optimization tool. Also, note that direct calculation of $\mathcal{G}$ is inefficient. We describe an efficient method in the sequel.

**Efficient Calculation of the Gram Matrix** Simple, straight-forward calculation of the Gram matrix is time consuming. To illustrate this, suppose each trial is of length $t_{end}^i = n$, then calculation of each basic block would take $\Theta(n^2)$ summation steps. We now describe a procedure based on dynamic-programming method for calculating the Gram matrix in a constant number of operations for each basic block.

Omitting the indexing over trials to ease notation, we are interested in calculating the basic block $\mathbf{K}_{tq}$. First, define $\mathbf{B}_{tq} = \sum_{k=1}^{t} k_{kq} \mathbf{A}^{t-k}$. the basic block $\mathbf{K}_{tq}$ can be recursively calculated in three different ways:

$$\mathbf{K}_{tq} = \mathbf{K}_{t(q-1)} \mathbf{A}^T + \mathbf{B}_{tq} \tag{5}$$

$$\mathbf{K}_{tq} = \mathbf{A} \mathbf{K}_{(t-1)q} + \left(\mathbf{B}_{qt}\right)^T \tag{6}$$

$$\mathbf{K}_{tq} = \mathbf{A} \mathbf{K}_{(t-1)(q-1)} \mathbf{A}^T + \left(\mathbf{B}_{qt}\right)^T + \mathbf{B}_{tq} - k_{tq} . \tag{7}$$

Thus, by adding Eq. (5) to Eq. (6) and subtracting Eq. (7) we get

$$\mathbf{K}_{tq} = \mathbf{A} \mathbf{K}_{(t-1)q} + \mathbf{K}_{t(q-1)} \mathbf{A}^T - \mathbf{A} \mathbf{K}_{(t-1)(q-1)} \mathbf{A}^T + k_{tq} I .$$

$\mathbf{B}_{tq}$ (and the entailed summation) is eliminated in exchange for a 2D dynamic program with initial conditions: $\mathbf{K}_{11} = k_{11} I$ , $K_{1q} = \mathbf{K}_{1(q-1)} \mathbf{A}^T + k_{1q} I$ , $\mathbf{K}_{t1} = \mathbf{A} \mathbf{K}_{(t-1)1} + k_{t1} I$.

**Table 1:** Mean $R^2$, MAE$_\varepsilon$ & MSE (across datasets, folds, hands and directions) for each algorithm.

| Algorithm | $R^2$ | | | MAE$_\varepsilon$ | | | MSE | | |
|---|---|---|---|---|---|---|---|---|---|
| | pos. | vel. | accl. | pos. | vel. | accl. | pos. | vel. | accl. |
| Kalman filter | 0.64 | 0.58 | 0.30 | 0.40 | 0.15 | 0.37 | 0.78 | 0.27 | 1.16 |
| DDT-linear | 0.59 | 0.49 | 0.17 | 0.63 | 0.41 | 0.58 | 0.97 | 0.50 | 1.23 |
| SVR-Spikernel | 0.61 | 0.64 | 0.37 | 0.44 | **0.14** | **0.34** | 0.76 | 0.20 | 0.98 |
| DDT-Spikernal | **0.73** | **0.67** | **0.40** | **0.37** | **0.14** | **0.34** | **0.50** | **0.16** | **0.91** |

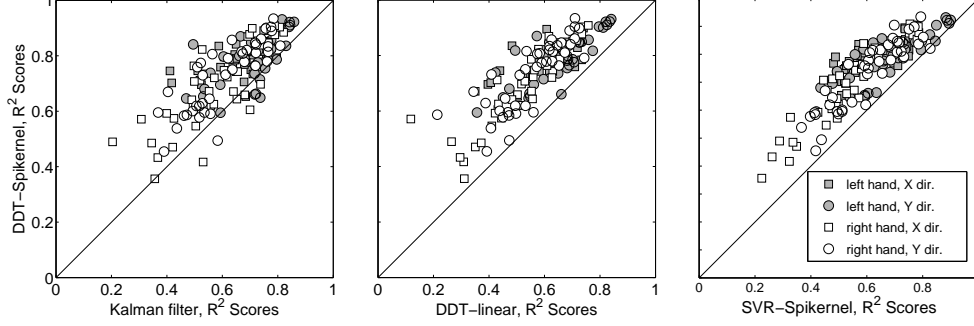

**Figure 1:** Correlation coefficients ($R^2$, of predicted and observed hand positions) comparisons of the DDT-Spikernel versus the Kalman filter (left), DDT-linear (center) and SVR-Spikernel (right). Each data point is the $R^2$ values obtained by the DDT-Spikernel and by another method in one fold of one of the datasets for one of the two axes of movement (circle / square) and one of the hands (filled/non-filled). Results above the diagonals are cases were the DDT-Spikernel outperformes.

**Suggested Optimization Method.** One possible way to solve the optimization problem (essentially, a modification of the method described in [4] for classification) is to sequentially solve a reduced problem with respect to a single constraint at a time. Define:

$$\delta_i = \left| \sum_j \left( \boldsymbol{\alpha}_j^* - \boldsymbol{\alpha}_j \right) \mathcal{G}_{ij} - y_i \right|_\varepsilon - \min_{\boldsymbol{\alpha}_i, \boldsymbol{\alpha}_i^* \in [0,c]} \left| \sum_j \left( \boldsymbol{\alpha}_j^* - \boldsymbol{\alpha}_j \right) \mathcal{G}_{ij} - y_i \right|_\varepsilon .$$

Then $\delta_i$ is the amount of $\varepsilon$-insensitive error that can be corrected for example $i$ by keeping all $\boldsymbol{\alpha}_{j \neq i}^{(*)}$ constant and changing $\boldsymbol{\alpha}_i^{(*)}$. Optimality is reached by iteratively choosing the example with the largest $\delta_i$ and changing its $\alpha_i^{(*)}$ within the $[0,c]$ limits to minimize the error for this example.

## 4 Experimental Setting

The data used in this work was recorded from the primary motor cortex of a Rhesus (Macaca Mulatta) monkey (˜4.5 kg). The monkey sat in a dark chamber, and up to 8 electrodes were introduced into MI area of each hemisphere. The electrode signals were amplified, filtered and sorted. The data used in this report was recorded on 8 different days and includes hand positions, sampled at 500Hz, spike times of single units (isolated by signal fit to a series of windows) and of multi units (detection by threshold crossing) sampled at $1ms$ precision. The monkey used two planar-movement manipulanda to control 2 cursors on the screen to perform a center-out reaching task. Each trial began when the monkey centered both cursors on a central circle. Either cursor could turn green, indicating the hand to be used in the trial. Then, one of eight targets appeared ('go signal'), the center circle disappeared and the monkey had to move and reach the target to receive liquid reward. The number of multi-unit channels ranged from 5 to 15, the number of single units was 20-27 and the average total was 34 units per dataset. The average spike rate per channel was 8.2 spikes/sec. More information on the recordings can be found in [9].

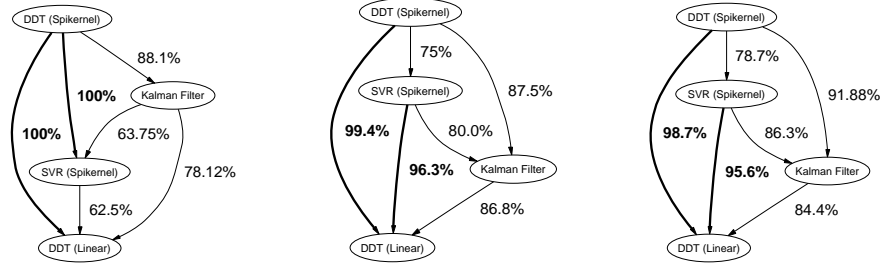

**Figure 2:** Comparison of $R^2$-performance between algorithms. Each algorithm is represented by a vertex. The weight of an edge between two algorithms is the fraction of tests in which the algorithm on top achieves higher $R^2$ score than the other. A bold edge indicates a fraction higher than $95\%$. Graphs from left to right are for position, velocity, and acceleration respectively.

The results that we present here refer to prediction of instantaneous hand movements during the period from 'Go Signal' to 'Target Reach' times of both hands in successful trials. Note that some of the trials required movement of the left hand while keeping the right hand steady and vise versa. Therefore, although we considered only movement periods of the trials, we had to predict both movement and non-movement for each hand. The cumulative time length of all the datasets was about 67 minutes. Since the correlation between the movements of the two hands tend to zero - we predicted movement for each hand separately, choosing the movement space to be $[x, y, v_x, v_y, a_x, a_y]^T$ for each of the hands (preliminary results using only $[x, y, v_x, v_y]^T$ were less accurate).

We preprocessed the spike trains into spike counts in a running window of $100ms$ (choice of window size is based on previous experience [11]). Hand position, velocity and acceleration were calculated using the 500Hz recordings. Both spike counts and hand movement were then sampled at steps of $100ms$ (preliminary results with step size $50ms$ were negligibly different for all algorithms). A labeled example $\{\mathbf{y}_t^i, \mathbf{o}_t^i\}$ for time $t$ in trial $i$ consisted of the previous 10 bins of population spike counts and the state, as a 6D vector for the left or right hand. Two such consecutive examples would than have 9 time bins of spike count overlap. For example, the number of cortical units $q$ in the first dataset was 43 (27 single and 16 multiple) and the total length of all the trials that were used in that dataset is 529 seconds. Hence in that session there are 5290 consecutive examples where each is a $43 \times 10$ matrix of spike counts along with two 6D vectors of end point movement.

In order to run our algorithm we had to choose base kernels, their parameters, $\mathbf{A}$ and $c$ (and $\theta$, to be introduced below). We used the Spikernel [11], a kernel designed to be used with spike rate patterns, and the simple dot product (i.e. linear regression). Kernel parmeters and $c$ were chosen (and subsequently held fixed) by 5 fold cross validation over half of the first dataset only. We compared DDT with the Spikernel and with the linear kernel to standard SVR using the Spikernel and the Kalman filter. We also obtained tracking results using both DDT and SVR with the standard exponential kernel. These results were slightly less accurate on average than with the Spikernel and are therefore omitted here. The Kalman filter was learned assuming the standard state space model ($\mathbf{y}_t = \mathbf{A}\mathbf{y}_{t-1} + \eta$ , $\mathbf{o}_t = \mathbf{H}\mathbf{y}_t + \xi$, where $\eta, \xi$ are white Gaussian noise with appropriate correlation matrices) such as in [16]. $\mathbf{y}$ belonged to the same 6D state space as described earlier. To ease the comparison - the same matrix $\mathbf{A}$ that was learned for the Kalman filter was used in our algorithm (though we show that it is not optimal for DDT), multiplied by a scaling parameter $\theta$. This parameter was selected to produce best *position* results on the training set. The selected $\theta$ value is $0.8$.

The figures that we show in Sec. 5 are of test results in 5 fold cross validation on the rest of the data. Each of the 8 remaining datasets was divided into 5 folds. 4/5 were used for

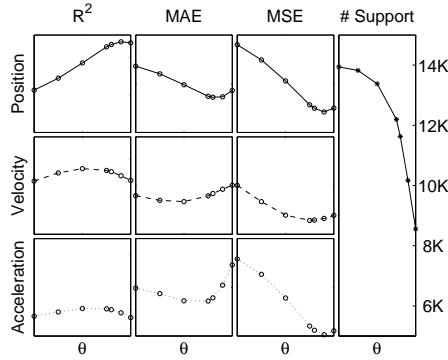

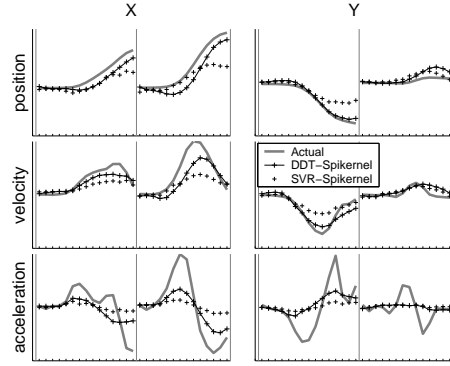

**Figure 3:** Effect of $\theta$ on $R^2$, MAE$_\varepsilon$ ,MSE and number of support vectors.

**Figure 4:** Sample of tracking with the DDT-Spikernel and the SVR-Spikernel.

training (with the parameters obtained previously and the remaining 1/5 as test set). This process was repeated 5 times for each hand. Altogether we had $8_{\text{sets}} \times 5_{\text{folds}} \times 2_{\text{hands}} = 80$ folds.

## 5   Results

We begin by showing average results across all datasets, folds, hands and X/Y directions for the four algorithms that are compared. Table. 1 shows mean Correlation Coefficients ($R^2$, between recorded and predicted movement values), Mean $\varepsilon$ insensitive Absolute Errors (MAE$_\varepsilon$) and Mean Square Errors (MSE). $R^2$ is a standard performance measure, MAE$_\varepsilon$ is the error minimized by DDT (subject to the regularization term) and MSE is minimized by the Kalman filter. Under all the above measures the DDT-Spikernel outperforms the rest with the SVR-Spikernel and the Kalman Filter alternating in second place.

To understand whether the performance differences are significant we look at the distribution of position (X and Y) $R^2$ values at each of the separate tests (160 altogether). Figure 1 shows scatter plots of $R^2$ results for position predictions. Each plot compares the DDT-Spikernel (on the Y axis) with one of the other three algorithms (on the X axes). It is clear that in spite large differences in accuracy across datasets, the algorithm pairs achieve similar success with the DDT-Spikernel achieving a better $R^2$ score in almost all cases.

To summarize the significance of $R^2$ differences we computed the number of tests in which one algorithm achieved a higher $R^2$ value than another algorithm (for all pairs, in each of the position, velocity and acceleration categories). The results of this tournament between the algorithms are presented in Figure 2 as winning percentages. The graphs produce a ranking of the algorithms and the percentages are the significances of the ranking between pairs. The DDT-Spikernel is significantly better then the rest in tracking position.

The matrix **A** in use is not optimal for our algorithm. The choice of $\theta$ scales its effect. When $\theta = 0$ we get the standard SVR algorithm (without state dynamics). To illustrate the effect of $\theta$ we present in Figure 3 the mean (over 5 folds, X/Y direction and hand) $R^2$ results on the first dataset as a function of $\theta$. It is clear that the value chosen to minimize position error is not optimal for minimizing velocity and acceleration errors. Another important effect of $\theta$ is the number of the support patterns in the learned model, which drops considerably (by about one third) when the effect of the dynamics is increased. This means that more training points fall strictly within the $\varepsilon$-tube in training, suggesting that the kernel which tacitly results from the dynamical model is better suited for the problem. Lastly, we show a sample of test tracking results for the DDT-Spikernel and SVR-Spikernel in Figure 4. Note that the acceleration values are not smooth and are, therefore, least aided by the dynamics of the model. However, adding acceleration to the model improves the prediction of position.

# 6    Conclusion

We described and reported experiments with a dynamical system that combines a linear state mapping with a nonlinear observation-to-state mapping. The estimation of the system's parameters is transformed to a dual representation and yields a novel kernel for temporal modelling. When a linear kernel is used, the DDT system has a similar form to the Kalman filter as $t \to \infty$. However, the system's parameters are set so as to minimize the regularized $\varepsilon$-insensitive $\ell_1$ loss between state trajectories. DDT also bares similarity to SVR, which employs the same loss yet without the state dynamics. Our experiments indicate that by combining a kernel-induced feature space, linear state dynamics, and using a robust loss we are able to leverage the trajectory prediction accuracy and outperform common approaches. Our next step toward an accurate brain-machine interface for predicting hand movements is the development of a learning procedure for the state dynamic mapping $\mathbf{A}$ and further developments of neurally motivated and compact representations.

**Acknowledgments**    This study was partly supported by a center of excellence grant (8006/00) administered by the ISF, BMBF-DIP, by the U.S. Israel BSF and by the IST Programme of the European Community, under the PASCAL Network of Excellence, IST-2002-506778. L.S. is supported by a Horowitz fellowship.

# References

[1] A. E. Brockwell, A. L. Rojas, and R. E. Kass. Recursive bayesian decoding of motor cortical signals by particle filtering. *Journal of Neurophysiology*, 91:1899–1907, 2004.

[2] E. N. Brown, L. M. Frank, D. Tang, M. C. Quirk, and M. A. Wilson. A statistical paradigm for neural spike train decoding applied to position prediction from ensemble firing patterns of rat hippocampal place cells. *Journal of Neuroscience*, 18(7411–7425), 1998.

[3] J. M. Carmena, M. A. Lebedev, R. E. Crist, J. E. O'Doherty, D. M. Santucci, D. F. Dimitrov, P. G. Patil, C. S. Henriques, and M. A. L. Nicolelis. Learning to control a brain-machine interface for reaching and grasping by primates. *PLOS Biology*, 1(2):001–016, 2003.

[4] K. Crammer and Y. Singer. On the algorithmic implementation of multiclass kernel-based vector machines. *Jornal of Machine Learning Research*, 2:265–292, 2001.

[5] J. Eichhorn, A. Tolias, A. Zien, M. Kuss, C. E. Rasmussen, J. Weston, N. Logothetis, and B. Schölkopf. Prediction on spike data using kernel algorithms. In *NIPS 16*. MIT Press, 2004.

[6] A. P. Georgopoulos, J. Kalaska, and J. Massey. Spatial coding of movements: A hypothesis concerning the coding of movement direction by motor cortical populations. *Experimental Brain Research (Supp)*, 7:327–336, 1983.

[7] R. E. Isaacs, D. J. Weber, and A. B. Schwartz. Work toward real-time control of a cortical neural prothesis. *IEEE Trans Rehabil Eng*, 8(196–198), 2000.

[8] C. Mehring, J. Rickert, E. Vaadia, S. C. de Oliveira, A. Aertsen, and S. Rotter. Inference of hand movements from local field potentials in monkey motor cortex. *Nature Neur.*, 6(12), 2003.

[9] R. Paz, T. Boraud, C. Natan, H. Bergman, and E. Vaadia. Preparatory activity in motor cortex reflects learning of local visuomotor skills. *Nature Neur.*, 6(8):882–890, August 2003.

[10] M. D. Serruya, N. G. Hatsopoulos, L. Paninski, M. R. Fellows, and J. P. Donoghue. Instant neural control of a movement signal. *Nature*, 416:141–142, March 2002.

[11] L. Shpigelman, Y. Singer, R. Paz, and E. Vaadia. Spikernels: Embedding spiking neurons in inner product spaces. In *NIPS 15*, Cambridge, MA, 2003. MIT Press.

[12] A. Smola and B. Scholkop. A tutorial on support vector regressio. In *NeuroCOLT2 Technical Report*, 1998.

[13] S. I. H. Tillery, D. M. Taylor, and A. B. Schwartz. Training in cortical control of neuroprosthetic devices improves signal extraction from small neuronal ensembles. *Reviews in the Neurosciences*, 14:107–119, 2003.

[14] V. Vapnik. *The Nature of Statistical Learning Theory*. Springer, N.Y., 1995.

[15] J. Wessberg, C. R. Stambaugh, J. D. Kralik, P. D. Beck, M. Laubach, J. K. Chapin, J. Kim, J. Biggs, M. A. Srinivasan, and M. A. Nicolelis. Real-time prediction of hand trajectory by ensembles of cortical neurons in primates. *Nature*, 408(16), November 2000.

[16] W. Wu, M. J. Black, Y. Gao, E. Bienenstock, M. Serruya, and J. P. Donoghue. Inferring hand motion from multi-cell recordings in motor cortex using a kalman filter. In *SAB02*, pages 66–73, Edinburgh, Scotland (UK), 2002.
